# Multi-task Gaussian Process Learning of Robot Inverse Dynamics

**Kian Ming A. Chai     Christopher K. I. Williams     Stefan Klanke     Sethu Vijayakumar**
School of Informatics, University of Edinburgh, 10 Crichton Street, Edinburgh EH8 9AB, UK
{k.m.a.chai, c.k.i.williams, s.klanke, sethu.vijayakumar}@ed.ac.uk

## Abstract

The inverse dynamics problem for a robotic manipulator is to compute the torques needed at the joints to drive it along a given trajectory; it is beneficial to be able to learn this function for adaptive control. A robotic manipulator will often need to be controlled while holding different loads in its end effector, giving rise to a multi-task learning problem. By placing independent Gaussian process priors over the latent functions of the inverse dynamics, we obtain a multi-task Gaussian process prior for handling multiple loads, where the inter-task similarity depends on the underlying inertial parameters. Experiments demonstrate that this multi-task formulation is effective in sharing information among the various loads, and generally improves performance over either learning only on single tasks or pooling the data over all tasks.

## 1   Introduction

The inverse dynamics problem for a robotic manipulator is to compute the torques $\boldsymbol{\tau}$ needed at the joints to drive it along a given trajectory, i.e. the motion specified by the joint angles $\boldsymbol{q}(t)$, velocities $\dot{\boldsymbol{q}}(t)$ and accelerations $\ddot{\boldsymbol{q}}(t)$, through time $t$. Analytical models for the inverse dynamics $\boldsymbol{\tau}(\boldsymbol{q}, \dot{\boldsymbol{q}}, \ddot{\boldsymbol{q}})$ are often infeasible, for example due to uncertainty in the physical parameters of the robot, or the difficulty of modelling friction. This leads to the need to *learn* the inverse dynamics.

A given robotic manipulator will often need to be controlled while holding different loads in its end effector. We refer to different loadings as different *contexts*. The inverse dynamics functions depend on the different contexts. A simple approach is to learn a different mapping for each context, but it is more attractive if one can exploit commonality in these related tasks to improve performance, i.e. to carry out *multi-task learning* (MTL) [1, 2]. The aim of this paper is to show how this can be carried out for the inverse dynamics problem using a multi-task Gaussian process (GP) framework.

In §2 we discuss the relevant theory for the problem. Details of how we optimize the hyperparameters of the multi-task GP are given in §3, and model selection is described in §4. Relationships to other work are discussed in §5, and the experimental setup and results are given in §6.

## 2   Theory

We first describe the relationship of inverse dynamics functions among contexts in §2.1. In §2.2 we review the multi-task GP regression model proposed in [3], and in §2.3 we describe how to derive a multi-task GP model for the inverse-dynamics problem.

### 2.1   Linear relationship of inverse dynamics between contexts

Suppose we have a robotic manipulator consisting of $J$ joints, and a set of $M$ loads. Figure 1 illustrates a six-jointed manipulator, with joint $j$ connecting links $j-1$ and $j$. We wish to learn the inverse

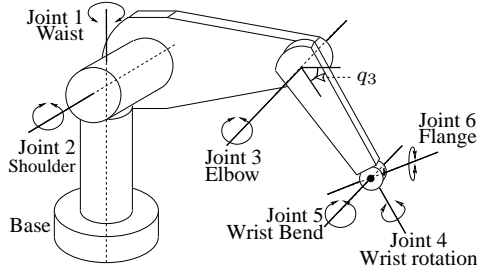

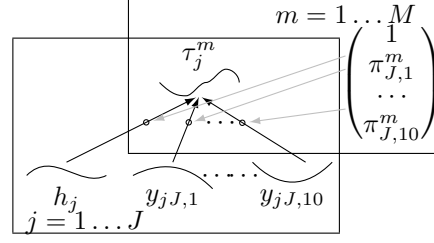

Figure 1: Schematic of the PUMA 560 without the end-effector (to be connected to joint 6).

Figure 2: A schematic diagram on how the different functions are related. A plate repeats its contents over the specified range.

dynamics model of the manipulator for the $m^{\text{th}}$ context, i.e. when it handles the $m^{\text{th}}$ load in its end-effector connected to the last link. We denote this by $\boldsymbol{\tau}^m(\boldsymbol{x}) \in \mathbb{R}^J$, with $\boldsymbol{x} \stackrel{\text{def}}{=} (\boldsymbol{q}^{\text{T}}, \dot{\boldsymbol{q}}^{\text{T}}, \ddot{\boldsymbol{q}}^{\text{T}})^{\text{T}} \in \mathbb{R}^{3J}$. It can be shown that the required torque for the $j^{\text{th}}$ joint can be written as [4]

$$\tau_j^m(\boldsymbol{x}) = \sum_{j'=j}^{J} \boldsymbol{y}_{jj'}^{\text{T}}(\boldsymbol{x}) \boldsymbol{\pi}_{j'}^m \qquad\qquad \boldsymbol{y}_{jj'} : \mathbb{R}^{3J} \mapsto \mathbb{R}^{10}, \qquad (1)$$

where the $\boldsymbol{y}_{jj'}$'s are vector-valued functions of $\boldsymbol{x}$, and $\boldsymbol{\pi}_{j'}^m \in \mathbb{R}^{10}$ is the vector of inertial parameters[1] of the $j'^{\text{th}}$ joint when manipulating the $m^{\text{th}}$ load. The inertial parameters for a joint depend on the physical characteristics of its corresponding link (e.g. mass) and are independent of $\boldsymbol{x}$.

When, as in our case, the loads are rigidly attached to the end effector, each load may be considered as part of the last link, and thus modifies the inertia parameters for the last link only [5]. The parameters for the other links remain unchanged since the parameters are local to the links and their frames. Denoting the common inertial parameters of the $j'^{\text{th}}$ link by $\boldsymbol{\pi}_{j'}^{\bullet}$, we can write

$$\tau_j^m(\boldsymbol{x}) = h_j(\boldsymbol{x}) + \boldsymbol{y}_{jJ}^{\text{T}}(\boldsymbol{x}) \boldsymbol{\pi}_J^m, \qquad\qquad \text{where } h_j(\boldsymbol{x}) \stackrel{\text{def}}{=} \sum_{j'=j}^{J-1} \boldsymbol{y}_{jj'}^{\text{T}}(\boldsymbol{x}) \boldsymbol{\pi}_{j'}^{\bullet}. \qquad (2)$$

Define $\tilde{\boldsymbol{y}}_j(\boldsymbol{x}) \stackrel{\text{def}}{=} (h_j(\boldsymbol{x}), (\boldsymbol{y}_{jJ}(\boldsymbol{x}))^{\text{T}})^{\text{T}}$ and $\tilde{\boldsymbol{\pi}}^m \stackrel{\text{def}}{=} (1, (\boldsymbol{\pi}_J^m)^{\text{T}})^{\text{T}}$, then $\tau_j^m(\boldsymbol{x}) = \tilde{\boldsymbol{y}}_j(\boldsymbol{x})^{\text{T}} \tilde{\boldsymbol{\pi}}^m$. Note that the $\tilde{\boldsymbol{y}}_j$s are shared among the contexts, while the $\tilde{\boldsymbol{\pi}}^m$s are shared among the $J$ links, as illustrated in Figure 2. This decomposition is not unique, since given a non-singular square $11 \times 11$ matrix $A_j$, setting $\boldsymbol{z}_j(\boldsymbol{x}) \stackrel{\text{def}}{=} A_j^{-\text{T}} \tilde{\boldsymbol{y}}_j(\boldsymbol{x})$ and $\boldsymbol{\rho}_j^m \stackrel{\text{def}}{=} A_j \tilde{\boldsymbol{\pi}}^m$, we also have

$$\tau_j^m(\boldsymbol{x}) = \tilde{\boldsymbol{y}}_j(\boldsymbol{x})^{\text{T}} A_j^{-1} A_j \tilde{\boldsymbol{\pi}}^m = \boldsymbol{z}_j(\boldsymbol{x})^{\text{T}} \boldsymbol{\rho}_j^m. \qquad (3)$$

Hence the vector of parameters $\tilde{\boldsymbol{\pi}}^\gamma$ is identifiable only up to a linear combination. Note that in general the matrix $A_j$ may vary across the joints.

## 2.2  Multi-task GP regression model

We give a brief summary of the multi-task Gaussian process (GP) regression model described in [3]. This model learns $M$ related functions $\{f^m\}_{m=1}^M$ by placing a zero mean GP prior which directly induces correlations between tasks. Let $t^m$ be the observation of the $m^{\text{th}}$ function at $\boldsymbol{x}$. Then the model is given by

$$\langle f^m(\boldsymbol{x}) f^{m'}(\boldsymbol{x}') \rangle \stackrel{\text{def}}{=} K_{mm'}^{\text{f}} k^{\text{x}}(\boldsymbol{x}, \boldsymbol{x}') \qquad t^m \sim \mathcal{N}(f^m(\boldsymbol{x}), \sigma_m^2), \qquad (4)$$

where $k^{\text{x}}$ is a covariance function over inputs, $K^{\text{f}}$ is a positive semi-definite (p.s.d) matrix of inter-task similarities, and $\sigma_m^2$ is the noise variance for the $m^{\text{th}}$ task.

## 2.3  Multi-task GP model for multiple contexts

We now show that the multi-task GP model can be used for inferring inverse dynamics for multiple contexts. We begin by placing independent zero mean GP priors on all the component functions of $\boldsymbol{z}_1(\cdot), \ldots, \boldsymbol{z}_J(\cdot)$. Let $\alpha$ be an index into the elements of the vector function $\boldsymbol{z}_j(\cdot)$, then our prior is

$$\langle z_{j\alpha}(\boldsymbol{x}) z_{j'\alpha'}(\boldsymbol{x}') \rangle = \delta_{jj'} \delta_{\alpha\alpha'} k_j^{\text{x}}(\boldsymbol{x}, \boldsymbol{x}'). \qquad (5)$$

In addition to independence specified by the Kronecker delta functions $\delta_{..}$, this model also imposes the constraint that all component functions for a given joint $j$ share the same covariance function $k_j^{\mathrm{x}}(\cdot, \cdot)$. With this prior over the $\boldsymbol{z}_j$s, the Gaussian process prior for $\tau_j^m(\cdot)$ is given by

$$\langle \tau_j^m(\boldsymbol{x}) \tau_{j'}^{m'}(\boldsymbol{x'}) \rangle = \delta_{jj'} (K_j^\rho)_{mm'} k_j^{\mathrm{x}}(\boldsymbol{x}, \boldsymbol{x'}), \qquad (6)$$

where we have set $\mathcal{P}_j \stackrel{\text{def}}{=} (\boldsymbol{\rho}_j^1 | \cdots | \boldsymbol{\rho}_j^M)$ and $K_j^\rho \stackrel{\text{def}}{=} \mathcal{P}_j^{\mathrm{T}} \mathcal{P}_j$, so that $(\boldsymbol{\rho}_j^m)^{\mathrm{T}} \boldsymbol{\rho}_j^{m'} = (K_j^\rho)_{mm'}$, the $(m, m')^{\text{th}}$ entry of the positive semi-definite matrix $K_j^\rho$. Notice that $K_j^\rho$ defines the similarity between different contexts. The rank of $K_j^\rho$ is the rank of $\mathcal{P}_j$, and is upper bounded by $\min(M, 11)$, reflecting the fact that there are at most 11 underlying latent functions (see Figure 2).

Let $t_j^m(\boldsymbol{x})$ be the observed value of $\tau_j^m(\boldsymbol{x})$. The deviations from $\tau_j^m(\boldsymbol{x})$ may be modelled with $t_j^m(\boldsymbol{x}) \sim \mathcal{N}(\tau_j^m(\boldsymbol{x}), (\sigma_j^m)^2)$, though in practice we let $\sigma_j \stackrel{\text{def}}{=} \sigma_j^1 \equiv \sigma_j^2 \ldots \equiv \sigma_j^M$, sharing the variance parameters among the contexts. This completes the correspondence with the multi-task GP model in eq. 4. Note, however, that in this case we have $J$ multi-task GP models, one for each joint.

This model is a simple and convenient one where the prior, likelihood and posterior factorize over joints. Hence inference and hyperparameter learning can be done separately for each joint.

**Making predictions** As in [3], inference in our model can be done by using the standard GP formulae for the mean and variance of the predictive distribution with the covariance function given in eq. 6 together with the normal noise model. The observations over all contexts for a given joint $j$ will be used to make the predictions. For the case of complete data (where there are observations at the same set of $\boldsymbol{x}$-values for all contexts) one can exploit the Kronecker-product structure [3, eq. 2].

### 2.3.1 The relationship among task similarity matrices

Let $\tilde{\Pi} \stackrel{\text{def}}{=} (\tilde{\boldsymbol{\pi}}^1 | \cdots | \tilde{\boldsymbol{\pi}}^M)$. Recall that $\tilde{\boldsymbol{\pi}}^m$ is an 11 dimensional vector. However, if the different loads in the end effector do not explore the full space (e.g. if some of the inertial parameters are constant over all loads), then it can happen that $s \stackrel{\text{def}}{=} \mathrm{rank}(\tilde{\Pi}) \leq \min(M, 11)$.

It is worthwhile to investigate the relationship between $K_j^\rho$ and $K_{j'}^\rho$, $j \neq j'$. Recall from eq. 3 that $\boldsymbol{\rho}_j^m \stackrel{\text{def}}{=} A_j \tilde{\boldsymbol{\pi}}^m$, where $A_j$ is a full-rank square matrix. This gives $\mathcal{P}_j = A_j \tilde{\Pi}$ and $K_j^\rho = \tilde{\Pi}^{\mathrm{T}} A_j^{\mathrm{T}} A_j \tilde{\Pi}$, so that $\mathrm{rank}(K_j^\rho) = \mathrm{rank}(\tilde{\Pi})$. Therefore the $K_j^\rho$s have the same rank for all joints, although their exact values may differ. This observation will be useful for model selection in §4.

## 3 Learning the hyperparameters — a staged optimization heuristic

In this section, we drop the joint index $j$ for the sake of brevity and clarity. The following applies separately for each joint. Let $\boldsymbol{t}^m$ be the vector of $n^m$ observed torques at the joint for context $m$, and $X^m$ be the corresponding $3J \times n^m$ design matrix. Further, let $X$ be the $3J \times N$ design matrix of distinct $\boldsymbol{x}$-configurations observed over all $M$ contexts. Given this data, we wish to optimize the marginal likelihood $L(\boldsymbol{\theta}^{\mathrm{x}}, K^\rho, \sigma^2) \stackrel{\text{def}}{=} p(\{\boldsymbol{t}^m\}_{m=1}^M | X, \boldsymbol{\theta}^{\mathrm{x}}, K^\rho, \sigma^2)$, where $\boldsymbol{\theta}^{\mathrm{x}}$ are the parameters of $k^{\mathrm{x}}$. As pointed out in [3], one may approach this either using general gradient-based optimization, or using expectation-maximization. In this paper, the former is used.

In general, the objective function $L(\boldsymbol{\theta}^{\mathrm{x}}, K^\rho, \sigma^2)$ will have multiple modes, and it is a difficult problem of how to locate the best mode. We propose a staged strategy during optimization to help localize the search region. This is outlined below, with details given in the subsections that follow.

---

**Require:** Starting positions $\boldsymbol{\theta}_0^{\mathrm{x}}$, $K_0^\rho$, $\sigma_0^2$, and rank $r$.

    {All $\arg\max$ operations are understood to find only the local maximum.}

1: Starting from $\boldsymbol{\theta}_0^{\mathrm{x}}$ and $\sigma_0^2$, find $(\boldsymbol{\theta}_1^{\mathrm{x}}, \sigma_1^2) = \arg\max_{\boldsymbol{\theta}^{\mathrm{x}}, \sigma^2} L(\boldsymbol{\theta}^{\mathrm{x}}, K_0^\rho, \sigma^2)$.

2: Calculate $K_1^\rho$ based on details in §3.2.

3: Starting from $\boldsymbol{\theta}_1^{\mathrm{x}}$, $K_1^\rho$, and $\sigma_0^2$, find $(\boldsymbol{\theta}_{\mathrm{ans}}^{\mathrm{x}}, K_{\mathrm{ans}}^\rho, \sigma_{\mathrm{ans}}^2) = \arg\max_{\boldsymbol{\theta}^{\mathrm{x}}, K^\rho, \sigma^2} L(\boldsymbol{\theta}^{\mathrm{x}}, K^\rho, \sigma^2)$.

---

The optimization order reflects the relative importance of the different constituents of the model. The most important is $k^{\mathrm{x}}$, hence the estimation of $\boldsymbol{\theta}^{\mathrm{x}}$ begins in step 1; the least important is $\sigma^2$, hence its estimation from the initial value $\sigma_0^2$ is in step 3. For our application, we find that this strategy works better than one which simultaneously optimizes for all the parameters.

### 3.1 The initial choice of $K^\rho$

The choice of $K_0^\rho$ is important, since it affects the search very early on. Reasonable values that admit ready interpretations are the matrix of ones $\mathbf{1}\mathbf{1}^\mathrm{T}$ and the identity matrix $I$. For $K_0^\rho = \mathbf{1}\mathbf{1}^\mathrm{T}$, we initially assume the contexts to be indistinguishable from each other; while for $K_0^\rho = I$, we initially assume the contexts to be independent given the kernel parameters, which is a multi-task learning model that has been previously explored, e.g. [6]. These two are at the opposite extremes in the spectrum of inter-context/task correlation, and we believe the merit of each will be application dependent. Since these two models have the same number of free parameters, we select the one with the higher likelihood as the starting point for the search in step 2. However, we note that in some applications there may be reasons to prefer one over the other.

### 3.2 Computation of $K_1^\rho$ in step 2

Given estimates $\boldsymbol{\theta}_1^\mathrm{x}$ and $\sigma_1^2$, we wish to estimate a $K_1^\rho$ from which the likelihood can be optimized in step 3. Here we give the sequence of considerations that leads to a formula for computing $K_1^\rho$.

Let $K_1^\mathrm{x}$ be the covariance matrix for all pairs in $X$, using $\boldsymbol{\theta}_1^\mathrm{x}$ for $k^\mathrm{x}$. Let $\mathcal{T}$ be an $N\times M$ matrix which corresponds to the true values of the torque function $\tau^m(\boldsymbol{x}_i)$ for $m = 1,\ldots,M$ and $i = 1,\ldots,N$. Then as per the EM step discussed in [3, eq. 4], we have

$$K_\mathrm{EM}^\rho = N^{-1}\left\langle \mathcal{T}^\mathrm{T}(K_1^\mathrm{x})^{-1}\mathcal{T}\right\rangle_{\tilde{\boldsymbol{\theta}}_0} \simeq N^{-1}\left\langle\mathcal{T}\right\rangle_{\tilde{\boldsymbol{\theta}}_0}^\mathrm{T}(K_1^\mathrm{x})^{-1}\left\langle\mathcal{T}\right\rangle_{\tilde{\boldsymbol{\theta}}_0}, \tag{7}$$

where the expectations are taken w.r.t a GP with parameters $\tilde{\boldsymbol{\theta}}_0 = (\boldsymbol{\theta}_1^\mathrm{x}, K_0^\rho, \sigma_1^2)$, and the $(i,m)^\mathrm{th}$ entry of $\langle\mathcal{T}\rangle_{\tilde{\boldsymbol{\theta}}_0}$ is the mean of $\tau^m(\boldsymbol{x}_i)$ with this GP. The approximation neglects the GP's variance; this is justifiable since the current aim is to obtain a starting estimate of $K^\rho$ for a search procedure.

There are two weaknesses with eq. 7 that we shall address. The first is that the rank of $\langle\mathcal{T}\rangle_{\tilde{\boldsymbol{\theta}}_0}$ is upper bounded by that of $K_0^\rho$, so that the rank of $K_\mathrm{EM}^\rho$ is similarly upper bounded.[2] This property is undesirable, particularly when $K_0^\rho = \mathbf{1}\mathbf{1}^T$. We ameliorate this by replacing $\langle\tau^m(\boldsymbol{x}_i)\rangle_{\tilde{\boldsymbol{\theta}}_0}$ with the corresponding observed value $t^m(\boldsymbol{x}_i)$ wherever it is available, and call the resultant matrix $\mathcal{T}_\mathrm{aug}$. The second weakness is that with the commonly used covariance functions, $K_1^\mathrm{x}$ will typically have rapidly decaying eigenvalues [7, §4.3.1]. To overcome this, we regularize its inversion by adding $\eta^2 I$ to the diagonal of $K_1^\mathrm{x}$ to give $K_\mathrm{aug}^\rho = N^{-1}\mathcal{T}_\mathrm{aug}^\mathrm{T}(K_1^\mathrm{x} + \eta^2 I)^{-1}\mathcal{T}_\mathrm{aug}$. We set $\eta^2$ to $\mathrm{tr}(\mathcal{T}_\mathrm{aug}^\mathrm{T}\mathcal{T}_\mathrm{aug})/(MN)$, so that $\mathrm{tr}(K_\mathrm{aug}^\rho) = M$ if $K_1^\mathrm{x}$ were the zero matrix.

Finally, the required $K_1^\rho$ is obtained from $K_\mathrm{aug}^\rho$ by constraining it to have rank $r$. This is currently achieved by computing the eigen-decomposition of $K_\mathrm{aug}^\rho$ and keeping only the top $r$ eigenvectors/values; it could also be implemented using an incomplete Cholesky decomposition.

### 3.3 Incorporating a novel task

Above we have assumed that data from all contexts is available at training time. However, we may encounter a new context for which we have not seen much data. In this case we fix $\boldsymbol{\theta}^\mathrm{x}$ and $\sigma^2$ while extending $K^\rho$ by an extra row and column for the new context, and it is only this new border which needs to be learned by maximising the marginal likelihood. Note that as $K^\rho$ is p.s.d this means learning only at most $M$ new parameters, or fewer if we exploit the rank-constraint property of $K^\rho$.

## 4 Model selection

The choice of the rank $r$ of $K_j^\rho$ in the model is important, since it reflects on the rank $s$ of $\tilde{\tilde{\Pi}}$. In our model, $r$ is not a hyperparameter to be optimized. Thus to infer its value we rely on an information criterion to select the most parsimonious correct model. Here, we use the Bayesian Information Criterion (BIC), but the use of Akaike or Hannan-Quinn criteria is similar.

Let $L_{jr}$ be the likelihood for each joint at optimized hyperparameters $\boldsymbol{\theta}_j^\mathrm{x}$, $K_j^\rho$, and $\sigma_j^2$, when $K_j^\rho$ is constrained to have rank $r$; let $n_j^m$ be the number of observations for the $j^\mathrm{th}$ joint in the $m^\mathrm{th}$

context, and $n \stackrel{\text{def}}{=} \sum_{j,m} n_j^m$ be the total number of observations; and let $d_j$ be the dimensionality of $\boldsymbol{\theta}_j^{\text{x}}$. Since the likelihood of the model factorizes over joints, we have

$$\text{BIC}(r) = -2\sum_{j=1}^{J} \log L_{jr} + \left(\sum_{j=1}^{J} d_j + \tfrac{J}{2}r(2M+1-r) + J\right)\log n, \qquad (8)$$

where $r(2M+1-r)/2$ is the number of parameters needed to define an incomplete Cholesky decomposition of rank $r$ for an $M \times M$ matrix. For selecting the appropriate rank of the $K_j^{\rho}$s, we compute and compare $\text{BIC}(r)$ for different values of $r$.

## 5   Relationships to other work

We consider related work first with regard to the inverse dynamics problem, and then to multi-task learning with Gaussian processes.

Learning methods for the single-context inverse dynamics problem can be found in e.g. [8], where the locally weighted projection regression (LWPR) method is used. Gaussian process methods for the same problem have also been shown to be effective [7, §2.5; 9]. The LWPR method has been extended to the multi-context situation by Petkos and Vijayakumar [5]. If the inertial parameters $\boldsymbol{\pi}_J^m$s are known for at least 11 contexts then the estimated torque functions can be used to estimate the underlying $\boldsymbol{y}_{jj'}$s using linear regression, and prediction in a novel context (with limited training data) will depend on estimating the inertial parameters for that context. Assuming the original estimated torque functions are imperfect, having more than 11 models for distinct known inertial parameters will improve load estimation. If the inertial parameters are unknown, the novel torque function can still be represented as a linear combination of a set of 11 linearly independent torque functions, and so one can estimate the inverse dynamics in a novel context by linear regression on those estimated functions. In contrast to the known case, however, no more than 11 models can be used [5, §V]. Another difference between known and unknown parameters is that in the former case the resulting $\boldsymbol{\pi}_J^m$s are interpretable, while in the latter there is ambiguity due to the $A_j$s in eq. 3.

Comparing our approach with [5], we note that: (a) their approach does not exploit the knowledge that the torque functions for the different contexts are known to share latent functions as in eq. 2, and thus it may be useful to learn the $M$ inverse dynamics models *jointly*. This is expected to be particularly advantageous when the data for each task explores rather different portions of $\boldsymbol{x}$-space; (b) rather than relying on least-squares methods (which assume equal error variances everywhere), our fully probabilistic model will propagate uncertainties (co-variances for jointly Gaussian models) automatically; and (c) eq. 6 shows that we do not need to be limited to exactly 11 reference contexts, either fewer or more than 11 can be used. On the other hand, using the LWPR methods will generally give rise to better computational scaling for large data-sets (although see approximate GP methods in [7, ch. 8]), and are perhaps less complex than the method in this paper.

Earlier work on multiple model learning such as Multiple Model Switching and Tuning (MMST) [10] uses an inverse dynamics model and a controller for each context, switching among the models to the one producing the most accurate predictions. The models are linear-in-the-parameters with known non-linear regressor functions of $\boldsymbol{x}$, and the number of models are assumed known. MMST involves very little dynamics learning, estimating only the linear parameters of the models. A closely related approach is Modular Selection and Identification for Control (MOSAIC) [11], which uses inverse dynamics models for control and forward dynamics models for context identification. However, MOSAIC was developed and tested on linear dynamics models without the insights into how eq. 1 may be used across contexts for more efficient and robust learning and control.

Early references to general multi-task learning are [1] and [2]. There has been a lot of work in recent years on MTL with e.g. neural networks, Dirichlet processes, Gaussian processes and support vector machines. Some previous models using GPs are summarized in [3]. An important related work is the semiparametric latent factor model [12] which has a number of latent processes which are linearly combined to produce observable functions as in eq. 3. However, in our model all the latent functions share a common covariance function, which reduces the number of free parameters and should thus help to reduce over-fitting. Also we note that the regression experiments by Teh et al. [12, §4] used a forward dynamics problem on a four-jointed robot arm for a single context, with an artificial linear mixing of the four target joint accelerations to produce six response variables. In contrast, we have shown how linear mixing arises naturally in a multi-context inverse dynamics situation. In relation

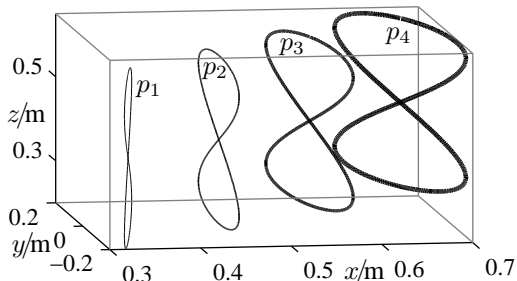

Figure 3: The four paths $p_1$, $p_2$, $p_3$, $p_4$. The robot base is located at $(0, 0, 0)$.

Table 1: The trajectories at which the training samples for each load are acquired. All loads have training samples from the common trajectory $(p_2, s_3)$. For the multiple-contexts setting, $c_{15}$, and hence $(p_4, s_4)$, is not used for training.

|       | $s_1$    | $s_2$    | $s_3$              | $s_4$      |
|-------|----------|----------|--------------------|------------|
| $p_1$ | $c_1$    | $c_7$    | $c_{13}$           | $c_{14}$   |
| $p_2$ | $c_6$    | $c_{12}$ | $c_1 \cdots c_{15}$| $c_5$      |
| $p_3$ | $c_{11}$ | $c_3$    | $c_4$              | $c_{10}$   |
| $p_4$ | $c_2$    | $c_8$    | $c_9$              | $c_{15}*$  |

Table 2: The average nMSEs of the predictions by LR and sGP, for joint 3 and for both kinds of test sets. Training set sizes given in the second row. The nMSEs are averaged over loads $c_1 \ldots c_{15}$.

|     | average nMSE for the interp$_m$ sets | | | | average nMSE for the extrap$_m$ sets | | | |
|-----|--------------------|--------------------|--------------------|--------------------|--------------------|--------------------|--------------------|--------------------|
|     | 20 | 170 | 1004 | 4000 | 20 | 170 | 1004 | 4000 |
| LR  | $1\times10^{-1}$ | $7\times10^{-4}$ | $6\times10^{-4}$ | $6\times10^{-4}$ | $5\times10^{-1}$ | $2\times10^{-1}$ | $2\times10^{-1}$ | $2\times10^{-1}$ |
| sGP | $1\times10^{-2}$ | $2\times10^{-7}$ | $2\times10^{-8}$ | $3\times10^{-9}$ | $1\times10^{-1}$ | $3\times10^{-2}$ | $4\times10^{-3}$ | $3\times10^{-3}$ |

to work by Bonilla et al. [3] described in section 2.2, we note that the factorization between inter-task similarity $K^{\mathrm{f}}$ and a common covariance function $k^{\mathrm{x}}$ is an *assumption* there, while we have shown that such decomposition is *inherent* in our application.

## 6 Experiments

**Data**  We investigate the effectiveness of our model with the Puma 560 (Figure 1), which has $J = 6$ degrees of freedom. We learn the inverse dynamic models of this robot manipulating $M = 15$ different loads $c_1, \ldots, c_{15}$ through four different figure-of-eight paths at four different speeds. The data for our experiments is obtained using a realistic simulation package [13], which models both Coulomb and viscous frictional forces. Figure 3 shows the paths $p_1, \ldots, p_4$ which are placed at 0.35m, 0.45m, 0.55m and 0.65m along the $x$-axis, at 0.36m, 0.40m, 0.44m and 0.48m along the $z$-axis, and rotated about the $z$-axis by $-10°$, $0°$, $10°$ and $20°$. There are four speeds $s_1, \ldots, s_4$, finishing a path in 20s, 15s, 10s and 5s respectively. In general, loads can have very different physical characteristics; in our case, this is done by representing each load as a cuboid with differing dimensions and mass, and attaching each load rigidly to a random point at the end-effector. The masses range evenly from 0.2kg for $c_1$ to 3.0kg for $c_{15}$; details of the other parameters are omitted due to space constraints.

For each load $c_m$, 4000 data points are sampled at regular intervals along the path for each path-speed (trajectory) combination $(p., s.)$. Each sample is the pair $(\boldsymbol{t}, \boldsymbol{x})$, where $\boldsymbol{t} \in \mathbb{R}^J$ are the observed torques at the joints, and $\boldsymbol{x} \in \mathbb{R}^{3J}$ are the joint angles, velocities and accelerations. This set of data is partitioned into train and test sets in the manner described below.

Acquiring training data combinatorially by sampling for every possible load-trajectory pair may be prohibitively expensive. One may imagine, however, that training data for the handling of a load can be obtained along a fixed reference trajectory $T_{\mathrm{r}}$ for calibration purposes, and also along a trajectory typical for that load, say $T_m$ for the $m^{\mathrm{th}}$ load. Thus, for each load, 2000 random training samples are acquired at a common reference trajectory $T_{\mathrm{r}} = (p_2, s_3)$, and an additional 2000 random training samples are acquired at a trajectory unique to each load; Table 1 gives the combinations. Therefore each load has a training set of 4000 samples, but acquired only on two different trajectories.

Following [14], two kinds of test sets are used to assess our models for (a) control along a repeated trajectory (which is of practical interest in industry), and (b) control along arbitrary trajectories (which is of general interest to roboticists). The test for (a) assesses the accuracy of torque predictions for staying *within* the trajectories that were used for training. In this case, the test set for load $c_m$, denoted by interp$_m$ for *interpolation*, consists of the rest of the samples from $T_{\mathrm{r}}$ and $T_m$ that are not used for training. The test for (b) assesses the accuracy also for *extrapolation* to trajectories not

sampled for training. The test set for this, denoted by $\mathsf{extrap}_m$, consists of all the samples that are not training samples for $c_m$.

In addition, we consider a data-poor scenario, and investigate the quality of the models using randomly selected subsets of the training data. The sizes of these subsets range from 20 to 4000.

**Results comparing GP with linear regression**    We first compare learning the inverse dynamics with Bayesian linear regression (LR) to learning with single-task Gaussian processes (sGP). For each context and each joint, we train a LR model and a sGP model with the corresponding training data separately. For LR, the covariates are $(\boldsymbol{x}, \mathbf{sgn}(\dot{\boldsymbol{q}}), 1)$, where $\mathbf{sgn}(\cdot)$ is the component-wise signum of its arguments; regression coefficients $\boldsymbol{\beta}$ and noise variance $\sigma^2$ are given a broad normal-inverse-gamma prior $p(\boldsymbol{\beta}, \sigma^2) \equiv \mathcal{N}(\boldsymbol{\beta}|\mathbf{0}, \sigma^2 \cdot 10^8 I)\mathcal{IG}(\sigma^2|1, 1)$, though note that the mean predictions do not depend on the parameters of the inverse-gamma prior on $\sigma^2$. The covariance function of each sGP model is a sum of an inhomogeneous linear kernel on $(\boldsymbol{x}, \mathbf{sgn}(\dot{\boldsymbol{q}}))$, a squared exponential kernel on $\boldsymbol{x}$, and an independent noise component [7, §4.2], with the first two using the automatic relevance determination parameterization [7, §5.1]. The hyperparameters of sGP are initialized by giving equal weightings among the covariates and among the components of the covariance function, and then learnt by optimizing the marginal likelihood independently for each context and each joint.

The trained LR and sGP models are used to predict torques for the $\mathsf{interp}_m$ and $\mathsf{extrap}_m$ data sets. For each test set, the normalized mean square error (nMSE) of the predictions are computed, by dividing the MSE by the variance of the test data. The nMSEs are then averaged over the 15 contexts for the $\mathsf{interp}_m$ and $\mathsf{extrap}_m$ tests. Table 2 shows how the averages for joint 3 vary with the number of training samples. Similar relative results are obtained for the other joints. The results show that sGP outperforms LR for both the test cases. As one would expect, the errors of LR level-off early at around 200 training samples, while the quality of predictions by sGP continues to improve with training sample size, especially so for the $\mathsf{interp}_m$ sets. Both sGP and LR do reasonably well on the $\mathsf{interp}_m$ sets, but not so well on the $\mathsf{extrap}_m$ sets. This suggests that learning from multiple contexts which have training data from different parts of the trajectory space will be advantageous.

**Results for multi-task GP**    We now investigate the merit of using MTL, using the training data tabulated in Table 1 for loads $c_1, \ldots, c_{14}$. We use $n$ to denote the number of observed torques for each joint totalled across the 14 contexts. Note that trajectory $(p_4, s_4)$ is entirely unobserved during learning, but is included in the $\mathsf{extrap}_m$ sets. We learn the hyperparameters of a multi-task GP model (mGP) for each joint by optimizing the marginal likelihood for all training data (accumulated across contexts) for that joint, as discussed in §3, using the same kernel and parameterization as for the sGP. This is done for ranks 2, 4, 5, 6, 8 and 10. Finally, a common rank $r$ for all the joints is chosen using the selection criterion given in §4. We denote the selected set of mGP models by mGP-BIC.

In addition to comparing with sGP, we also compare mGP-BIC with two other naïve schemes: (a) denoted by iGP, a collection of independent GPs for the contexts, but sharing kernel parameters of $k_j^{\mathsf{x}}$ among the contexts; and (b) denoted by pGP, a single GP for each joint that learns by pooling all training data from all the contexts. The iGP and pGP models can be seen as restrictions of the multi-task GP model, restricting $K_j^\rho$ to the identity matrix $I$ and the matrix of ones $\mathbf{1}\mathbf{1}^{\mathrm{T}}$ respectively.

As discussed in §3, the hyperparameters for the mGPs are initialized to either those of pGP or those of iGP during optimization, choosing the one with the higher marginal likelihood. For our data, we find that the choice is mostly iGP; pGP is only chosen for the case of joint 1 and $n < 532$. In addition, the chosen ranks based on the BIC are $r = 4$ for all cases of $n$, except for $n = 476$ and $n = 1820$ when $r = 5$ is selected instead.

Figure 4 gives results of sGP, iGP, pGP and mGP-BIC for both the $\mathsf{interp}_m$ and $\mathsf{extrap}_m$ test sets, and for joints 1 and 4. Plots for the other joints are omitted due to space constraints, but they are qualitatively similar to the plots for joint 4. The plots are the average nMSEs over the 14 contexts against $n$. The vertical scales of the plots indicate that extrapolation is at least an order of magnitude harder than interpolation. Since the training data are subsets selected independently for the different values of $n$, the plots reflect the underlying variability in sampling. Nevertheless, we can see that mGP-BIC performs favorably in almost all the cases, and especially so for the extrapolation task. For joint 1, we see a close match between the predictive performances of mGP-BIC and pGP, with mGP-BIC slightly better than pGP for the interpolation task. This is due to the limited variation among observed torques for this joint across the different contexts for the range of end-effector

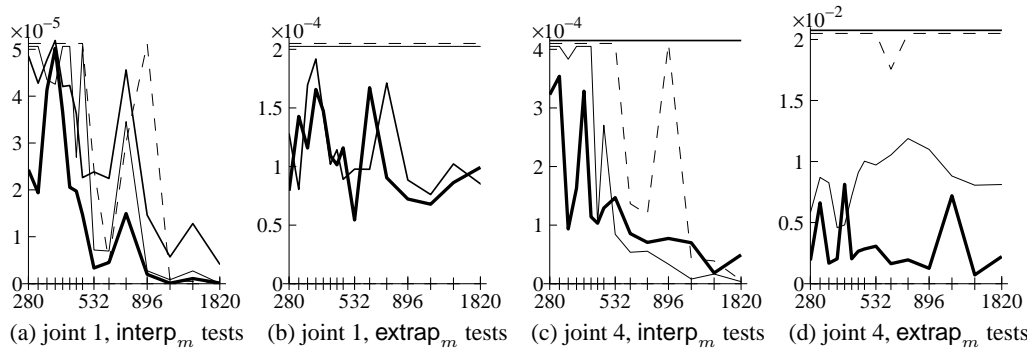

(a) joint 1, interp$_m$ tests    (b) joint 1, extrap$_m$ tests    (c) joint 4, interp$_m$ tests    (d) joint 4, extrap$_m$ tests

Figure 4: Average nMSEs of sGP ($--$), iGP (—), pGP (—) and mGP-BIC (—) against $n$ (on $\log_2$ scale). Ticks on the $x$-axes represent specified values of $n$. The vertical scales of the plots varies. A value above the upper limit of its vertical range is plotted with a nominal value near the top instead.

movements investigated here. Therefore it is not surprising that pGP produces good predictions for joint 1. For the other joints, iGP is usually the next best after mGP-BIC. In particular, iGP is better than sGP, showing that (in this case) combining all the data to estimate the parameters of a single common covariance function is better than separating the data to estimate the parameters of 14 covariance functions.

## 7 Summary

We have shown how the structure of the multiple-context inverse dynamics problem maps onto a multi-task GP prior as given in eq. 6, how the corresponding marginal likelihood can be optimized effectively, and how the rank of the $K_j^\rho$s can be chosen. We have demonstrated experimentally that the results of the multi-task GP method (mGP) are generally superior to sGP, iGP and pGP. Therefore it is advantageous to learn inverse dynamics models jointly using mGP-BIC, especially when each context/task explores different portions of the data space, a common case in dynamics learning. In future work we would like to investigate if coupling learning over joints is beneficial.

### Acknowledgments

We thank Sam Roweis for suggesting pGP as a baseline. This work is supported in part by the EU PASCAL2 ICT Programme, and in part by the EU FP6 SENSOPAC project grant to SV and SK. KMAC would also like to thank DSO NL for financial support.

## Footnotes

[1]We may also formulate our model using the more general vector of dynamic parameters which includes also the friction parameters, motor inertia etc. However, these additional parameters are independent of the load, and so can be absorbed into the function $h_j$ in eq. 2.

[2]This is not due to our approximation; indeed, it can be shown that the rank of $K_\mathrm{EM}^\rho$ is upper bounded by that of $K_0^\rho$ even if the exact EM update in eq. 7 has been used.

### References

[1] R. Caruana. Multitask Learning. *Machine Learning*, 28(1), July 1997.

[2] S. Thrun and L. Pratt, editors. *Learning to Learn*. Kluwer Academic Publishers, 1998.

[3] E. Bonilla, K. M. A. Chai, and C. K. I. Williams. Multi-task Gaussian Process Prediction. *NIPS 20*, 2008.

[4] L. Sciavicco and B. Siciliano. *Modelling and Control of Robot Manipulators*. Springer, 2000.

[5] G. Petkos and S. Vijayakumar. Load estimation and control using learned dynamics models. *IROS*, 2007.

[6] T. P. Minka and R. W. Picard. Learning How to Learn is Learning with Point Sets, 1997. URL `http://research.microsoft.com/~minka/papers/point-sets.html`. revised 1999.

[7] C. E. Rasmussen and C. K. I. Williams. *Gaussian Processes for Machine Learning*. MIT Press, 2006.

[8] S. Vijayakumar and S. Schaal. LWPR: An $O(n)$ Algorithm for Incremental Real Time Learning in High Dimensional Space. *ICML 2000*, 2000.

[9] D. Nguyen-Tuong, J. Peters, and M. Seeger. Computed torque control with nonparametric regression models. *ACC 2008*, 2008.

[10] M. Kemal Cılız and K. S. Narendra. Adaptive control of robotic manipulators using multiple models and switching. *Int. J. Rob. Res.*, 15(6):592–610, 1996.

[11] M. Haruno, D. M. Wolpert, and M. Kawato. MOSAIC Model for Sensorimotor Learning and Control. *Neural Comp.*, 13(10):2201–2220, 2001.

[12] Y. W. Teh, M. Seeger, and M. I. Jordan. Semiparametric latent factor models. *10th AISTATS*, 2005.

[13] P. I. Corke. A robotics toolbox for MATLAB. *IEEE Rob. and Auto. Magazine*, 3(1):24–32, 1996.

[14] E. Burdet and A. Codourey. Evaluation of parametric and nonparametric nonlinear adaptive controllers. *Robotica*, 16(1):59–73, 1998.
